# A Neural Autoregressive Topic Model

**Hugo Larochelle**
Département d'informatique
Université de Sherbrooke
*hugo.larochelle@usherbrooke.ca*

**Stanislas Lauly**
Département d'informatique
Université de Sherbrooke
*stanislas.lauly@usherbrooke.ca*

## Abstract

We describe a new model for learning meaningful representations of text documents from an unlabeled collection of documents. This model is inspired by the recently proposed Replicated Softmax, an undirected graphical model of word counts that was shown to learn a better generative model and more meaningful document representations. Specifically, we take inspiration from the conditional mean-field recursive equations of the Replicated Softmax in order to define a neural network architecture that estimates the probability of observing a new word in a given document given the previously observed words. This paradigm also allows us to replace the expensive softmax distribution over words with a hierarchical distribution over paths in a binary tree of words. The end result is a model whose training complexity scales logarithmically with the vocabulary size instead of linearly as in the Replicated Softmax. Our experiments show that our model is competitive both as a generative model of documents and as a document representation learning algorithm.

## 1 Introduction

In order to leverage the large amount of available unlabeled text, a lot of research has been devoted to developing good probabilistic models of documents. Such models are usually embedded with latent variables or topics, whose role is to capture salient statistical patterns in the co-occurrence of words within documents.

The most popular model is latent Dirichlet allocation (LDA) [1], a directed graphical model in which each word is a sample from a mixture of global word distributions (shared across documents) and where the mixture weights vary between documents. In this context, the word multinomial distributions (mixture components) correspond to the topics and a document is represented as the parameters (mixture weights) of its associated distribution over topics. Once trained, these topics have been found to extract meaningful groups of semantically related words and the (approximately) inferred topic mixture weights have been shown to form a useful representation for documents.

More recently, Salakhutdinov and Hinton [2] proposed an alternative undirected model, the Replicated Softmax which, instead of representing documents as distributions over topics, relies on a binary distributed representation of the documents. The latent variables can then be understood as topic features: they do not correspond to normalized distributions over words, but to unnormalized factors over words. A combination of topic features generates a word distribution by multiplying these factors and renormalizing. They show that the Replicated Softmax allows for very efficient inference of a document's topic feature representation and outperforms LDA both as a generative model of documents and as a method for representing documents in an information retrieval setting.

While inference of a document representation is efficient in the Replicated Softmax, one of its disadvantages is that the complexity of its learning update scales linearly with the vocabulary size $V$, i.e. the number of different words that are observed in a document. The factor responsible for this

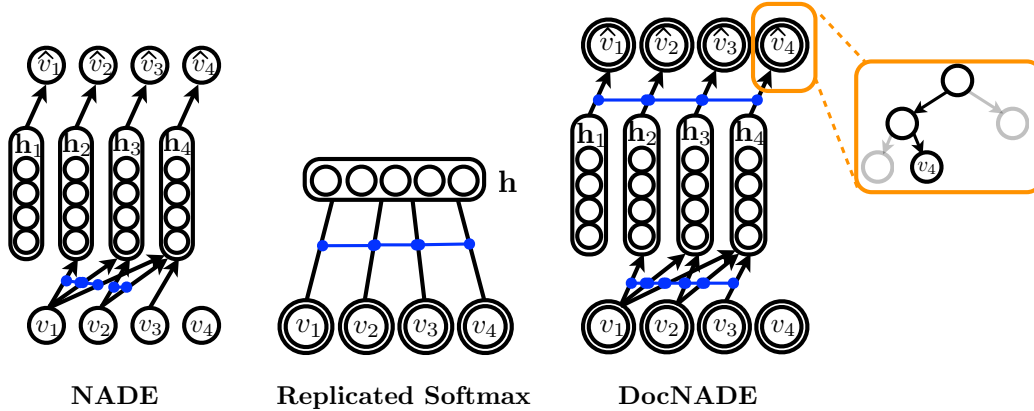

Figure 1: **(Left)** Illustration of NADE. Colored lines identify the connections that share parameters and $\widehat{v}_i$ is a shorthand for the autoregressive conditional $p(v_i|\mathbf{v}_{<i})$. The observations $v_i$ are binary. **(Center)** Replicated Softmax model. Each multinomial observation $v_i$ is a word. Connections between each multinomial observation $v_i$ and hidden units are shared. **(Right)** DocNADE, our proposed model. Connections between each multinomial observation $v_i$ and hidden units are also shared, and each conditional $p(v_i|\mathbf{v}_{<i})$ is decomposed into a tree of binary logistic regressions.

complexity is the conditional distribution of the words given the latent variables, which corresponds to a $V$-way multinomial logistic regression. In a realistic application scenario, $V$ will usually be in the 100 000's.

The Replicated Softmax is in fact a generalization of the restricted Boltzmann machine (RBM). The RBM is an undirected graphical model with binary observed and latent variables organized in a bi-partite graph. The Replicated Softmax instead has multinomial (softmax) observed variables and shares (replicates) across all observed variables the parameters between an observed variable and all latent variables.

A good alternative to the RBM is the neural autoregressive distribution estimator (NADE) [3]. It is similar to an autoencoder neural network, in that it takes as input a vector of observations and outputs a vector of the same size. However, the connectivity of NADE has been specifically chosen so as to make it a proper generative model for vectors of binary observations. More specifically, NADE outputs the conditional probabilities of each observation given the other observations to its left in the vector. Taking the product of all these conditional probabilities thus yields a proper joint probability over the whole input vector of observations. One advantage of NADE is that computing the parameter gradient of the data negative log-likelihood requires no approximation (unlike in an RBM). Also, unlike in the RBM, NADE does not require a symmetric connectivity, i.e. the weights going in and out of its hidden units can be different.

In this work, we describe DocNADE, a neural network topic model that is similarly inspired by the Replicated Softmax. From the Replicated Softmax, we derive an efficient approach for computing the hidden units of the network. As for the computation of the distribution of words given the hidden units, our feed-forward neural network approach leaves us free to use other conditionals than the $V$-way multinomial logistic regression implied by the Replicated Softmax. In particular, we instead opt for a hierarchy of binary logistic regressions, organized in a binary tree where each leaf corresponds to a word of the vocabulary. This allows us to obtain a complexity of computing the probability of an observed word scaling sublinearly with $V$. Our experiments show that DocNADE is competitive both as a generative model of documents and as a learning algorithm for extracting meaningful representations of documents.

## 2 Neural Autoregressive Distribution Estimation

We start with the description of the original NADE. NADE is a generative model over vectors of binary observations $\mathbf{v} \in \{0, 1\}^D$. Through the probability chain rule, it decomposes $p(\mathbf{v}) = \prod_{i=1}^{D} p(v_i|\mathbf{v}_{<i})$ and computes all $p(v_i|\mathbf{v}_{<i})$ using the feed-forward architecture

$$\mathbf{h}_i(\mathbf{v}_{<i}) = \mathrm{sigm}\left(\mathbf{c} + \mathbf{W}_{:,<i}\mathbf{v}_{<i}\right), \qquad p(v_i = 1|\mathbf{v}_{<i}) = \mathrm{sigm}\left(b_i + \mathbf{V}_{i,:}\mathbf{h}_i(\mathbf{v}_{<i})\right) \qquad (1)$$

for $i \in \{1, \ldots, D\}$, where $\mathrm{sigm}(x) = 1/(1 + \exp(-x))$, $\mathbf{W} \in \mathbb{R}^{H \times D}$ and $\mathbf{V} \in \mathbb{R}^{D \times H}$ are connection parameter matrices, $\mathbf{b} \in \mathbb{R}^D$ and $\mathbf{c} \in \mathbb{R}^H$ are bias parameter vectors, $\mathbf{v}_{<i}$ is the subvector $[v_1, \ldots, v_{i-1}]^\top$ and $\mathbf{W}_{:,<i}$ is a matrix made of the $i-1$ first columns of $\mathbf{W}$.

This architecture corresponds to a neural network with several parallel $\mathbf{h}_i(\mathbf{v}_{<i})$ hidden layers and tied weighted connections between $v_i$ and each hidden unit $h_{ij}(\mathbf{v}_{<i})$. Figure 1 gives an illustration. Though each $p(v_i = 1|\mathbf{v}_{<i})$ requires the computation of its own hidden layer $\mathbf{h}_i(\mathbf{v}_{<i})$, the tied weights allows to compute them all in $O(DH)$, where $H$ is the size of each hidden layer $\mathbf{h}_i(\mathbf{v}_{<i})$.

Equation 1 provides all the necessary conditionals to compute $p(\mathbf{v}) = \prod_i p(v_i|\mathbf{v}_{<i})$. The parameters $\{\mathbf{b}, \mathbf{c}, \mathbf{W}, \mathbf{V}\}$ can then be learned by minimizing the negative log-likelihood with stochastic gradient descent.

The connectivity behind NADE (i.e. the presence of a separate hidden layer $\mathbf{h}_i(\mathbf{v}_{<i})$ for each $p(v_i = 1|\mathbf{v}_{<i})$ with weight sharing) were directly inspired from the RBM. An RBM is an undirected graphical model in which latent binary variables $\mathbf{h}$ interact with the observations $\mathbf{v}$ through an energy function $E(\mathbf{v}, \mathbf{h})$, converted into a distribution over $\mathbf{v}$ as follows:

$$E(\mathbf{v}, \mathbf{h}) = -\mathbf{h}^\top \mathbf{W} \mathbf{v} - \mathbf{b}^\top \mathbf{v} - \mathbf{c}^\top \mathbf{h}, \qquad p(\mathbf{v}) = \sum_\mathbf{h} \exp(-E(\mathbf{v}, \mathbf{h}))/Z, \qquad (2)$$

where $Z$ is known as the partition function and ensures that $p(\mathbf{v})$ is a valid distribution and sums to 1. Computing the conditional $p(v_i = 1|\mathbf{v}_{<i})$ in an RBM is generally intractable but can be approximated through mean-field inference. Mean-field inference approximates the full conditional $p(v_i, \mathbf{v}_{>i}, \mathbf{h}|\mathbf{v}_{<i})$ as a product of independent Bernoulli distributions $q(v_k = 1|\mathbf{v}_{<i}) = \mu_k(i)$ and $q(h_j = 1|\mathbf{v}_{<i}) = \tau_j(i)$. To find the values of the variational parameters $\mu_k(i), \tau_j(i)$ that minimize the KL-divergence with $p(v_i, \mathbf{v}_{>i}, \mathbf{h}|\mathbf{v}_{<i})$, the following message passing equations are applied until convergence, for $k \in \{i, \ldots, D\}$ and $j \in \{1, \ldots, H\}$ (see Larochelle and Murray [3] for the derivation):

$$\tau_j(i) \leftarrow \mathrm{sigm}\left( c_j + \sum_{k \geq i} W_{jk} \mu_k(i) + \sum_{k < i} W_{jk} v_k \right), \quad \mu_k(i) \leftarrow \mathrm{sigm}\left( b_k + \sum_j W_{jk} \tau_j(i) \right). \tag{3}$$

The variational parameter $q(v_i = 1|\mathbf{v}_{<i}) = \mu_i(i)$ can then be used to approximate $p(v_i = 1|\mathbf{v}_{<i})$.

NADE is derived from the application of each message passing equation only once (with $\mu_j(i)$ initialized to 0), but compensates by untying the weights between each equation and training the truncation directly to fit the available data. The end result is thus the feed-forward architecture of Equation 1.

The relationship between the RBM and NADE is important, as it specifies an effective way of sharing the hidden layer parameters across the conditionals $p(v_i = 1|\mathbf{v}_{<i})$. In fact, other choices not inspired by the RBM have proven less successful (see Bengio and Bengio [4] and Larochelle and Murray [3] for a discussion).

## 3 Replicated Softmax

Documents can't be easily modeled by the RBM for two reasons: words are not binary but multinomial observations and documents may contain a varying number of words. An observation vector $\mathbf{v}$ is now a sequence of words indices $v_i$ taking values in $\{1, \ldots, V\}$, while the size $D$ of $\mathbf{v}$ can vary.

To address these issues, Salakhutdinov and Hinton [2] proposed the Replicated Softmax model, which uses the following energy function

$$E(\mathbf{v}, \mathbf{h}) = -D\, \mathbf{c}^\top \mathbf{h} + \sum_{i=1}^{D} -\mathbf{h}^\top \mathbf{W}_{:,v_i} - b_{v_i} = -D\, \mathbf{c}^\top \mathbf{h} - \mathbf{h}^\top \mathbf{W} \mathbf{n}(\mathbf{v}) - \mathbf{b}^\top \mathbf{n}(\mathbf{v}), \qquad (4)$$

where $\mathbf{W}_{:,v_i}$ is the $v_i^{\text{th}}$ column vector of matrix $\mathbf{W}$ and $\mathbf{n}(\mathbf{v})$ is a vector of size $V$ containing the word count of each word in the vocabulary. Notice that this energy shares its connection parameters across different positions $i$ in $\mathbf{v}$. Figure 1 provides an illustration. Notice also that the larger $\mathbf{v}$ is,

the more important the terms summed over $i$ in the energy will be. Hence, the hidden bias term $\mathbf{c}^\top \mathbf{h}$ is multiplied by $D$ to maintain a certain balance between all terms.

In this model, the conditional across layers $p(\mathbf{v}|\mathbf{h}) = \prod_{i=1}^{D} p(v_i|\mathbf{h})$ and $p(\mathbf{h}|\mathbf{v}) = \prod_j p(h_j|\mathbf{v})$ factorize and are such that:

$$p(h_j = 1|\mathbf{v}) = \text{sigm}(Dc_j + \sum_i W_{jv_i}) \qquad p(v_i = w|\mathbf{h}) = \frac{\exp(b_w + \mathbf{h}^\top \mathbf{W}_{:,w})}{\sum_{w'} \exp(b_{w'} + \mathbf{h}^\top \mathbf{W}_{:,w'})} \qquad (5)$$

The normalized exponential in $p(v_i = w|\mathbf{h})$ is known as the softmax nonlinearity. We see that, given a value of the topic features $\mathbf{h}$, the distribution each word $v_i$ in the document can be understood as the normalized product of multinomial topic factors $\exp(h_j \mathbf{W}_{j,:}^\top)$ and $\exp(\mathbf{b})$, as opposed to a mixture of multinomial topic distributions.

The gradient of the negative log-likelihood of a single training document $\mathbf{v}^t$ with respect to any parameter $\theta$ has the simple form

$$\frac{\partial - \log p(\mathbf{v}^t)}{\partial \theta} = \mathbb{E}_{\mathbf{h}|\mathbf{v}^t} \left[ \frac{\partial}{\partial \theta} E(\mathbf{v}^t, \mathbf{h}) \right] - \mathbb{E}_{\mathbf{v},\mathbf{h}} \left[ \frac{\partial}{\partial \theta} E(\mathbf{v}, \mathbf{h}) \right]. \qquad (6)$$

Computing the last expectation exactly is too expensive, hence the contrastive divergence [5] approximation is used: the expectation over $\mathbf{v}$ is replaced by a point estimate at a so-called "negative" sample, obtained from $K$ steps of blocked Gibbs sampling based on Equation 5 initialized at $\mathbf{v}^t$. Once a negative sample is obtained Equation 6 can be estimated and used with stochastic gradient descent training.

Unfortunately, computing $p(v_i = w|\mathbf{h})$ to sample the words during Gibbs sampling is linear in $V$ and $H$, where $V$ tends to be quite large. Fortunately, given $\mathbf{h}$, it needs to be computed only once before sampling all $D$ words in $\mathbf{v}$. However, when $\mathbf{h}$ is re-sampled, $p(v_i = w|\mathbf{h})$ must be recomputed. Hence, the computation of $p(v_i = w|\mathbf{h})$ is usually the most expensive component of the learning update: sampling the hidden layer given $\mathbf{v}$ is only in $O(DH)$, and repeatably sampling from the softmax multinomial distribution can be in $O(V)$. This makes for a total complexity in $O(KVH + DH)$ of the learning update.

## 4 Document NADE

More importantly for the context of this paper, it can be shown that mean-field inference of $p(v_i = w|\mathbf{v}_{<i})$ in the Replicated Softmax corresponds to the following message passing equations, for $k \in \{i, \dots, D\}$, $j \in \{1, \dots, H\}$ and $w \in \{1, \dots, V\}$:

$$\tau_j(i) \leftarrow \text{sigm}\left( D\, c_j + \sum_{k \geq i} \sum_{w'=1}^{V} W_{jw'} \mu_{kw'}(i) + \sum_{k<i} W_{jv_k} \right), \qquad (7)$$

$$\mu_{kw}(i) \leftarrow \frac{\exp(b_w + \sum_j W_{jw} \tau_j(i))}{\sum_{w'} \exp(b_{w'} + \sum_j W_{jw'} \tau_j(i))}. \qquad (8)$$

Following the derivation of NADE, we can truncate the application of these equations to obtain a feed-forward architecture providing an estimate of $p(v_i = w|\mathbf{v}_{<i})$ through $\mu_{iw}(i)$ for all $i$. Specifically, if we consider a single iteration of message passing with $\mu_{kw'}(i)$ initialized to 0, we untie the parameter weight matrix between each equation into two separate matrices $\mathbf{W}$ and $\mathbf{V}$ and remove the multiplication by $D$ of the hidden bias, we obtain the following feed-forward architecture:

$$\mathbf{h}_i(\mathbf{v}_{<i}) = \text{sigm}\left( \mathbf{c} + \sum_{k<i} \mathbf{W}_{:,v_k} \right), \quad p(v_i = w|\mathbf{v}_{<i}) = \frac{\exp(b_w + \mathbf{V}_{w,:}\mathbf{h}_i(\mathbf{v}_{<i}))}{\sum_{w'} \exp(b_{w'} + \mathbf{V}_{w',:}\mathbf{h}_i(\mathbf{v}_{<i}))} \qquad (9)$$

for $i \in \{1, \dots, D\}$. In words, the probability of the $i^{\text{th}}$ word $v_i$ is based on a position dependent hidden layer $\mathbf{h}_i(\mathbf{v}_{<i})$ which extracts a representation out of all previous words $\mathbf{v}_{<i}$. This latent representation is efficient to compute, as it consists simply in a linear transformation followed by an element-wise sigmoidal nonlinearity. Unlike in the Replicated Softmax, we have found that multiplying the hidden bias by $D$ was not necessary and, in fact, slightly hampered the performance of the model, so we opted for its removal.

To obtain the probability of the next word $v_{i+1}$, one must first compute the hidden layer

$$\mathbf{h}_{i+1}(\mathbf{v}_{<i+1}) = \text{sigm}(\mathbf{c} + \sum_{k<i+1} W_{:,v_k}) = \text{sigm}(\mathbf{W}_{:,v_i} + \mathbf{c} + \sum_{k<i} \mathbf{W}_{:,v_k}) \qquad (10)$$

which is efficiently computed by reusing the previous linear transformation $\mathbf{c} + \sum_{k<i} \mathbf{W}_{:,v_k}$ and adding $\mathbf{W}_{:,v_i}$. With this procedure, we see that computing all hidden layers $\mathbf{h}_i(\mathbf{v}_{<i})$ is in $O(DH)$.

Computing the softmax nonlinearity of each $p(v_i = w|\mathbf{v}_{<i})$ in Equation 9 requires time linear in $V$, which we would like to avoid. Fortunately, unlike in the Replicated Softmax, we are not tied to the use of a large softmax nonlinearity to model probabilities over words. In the literature on neural probabilistic language models, the large softmax over words is often replaced by a probabilistic tree model in which each path from the root to a leaf corresponds to a word [6, 7]. The probabilities of each left/right transitions in the tree are modeled by a set of binary logistic regressors and the probability of a given word is then obtained by multiplying the probabilities of each left/right choice of the associated tree path.

Specifically, let $\mathbf{l}(v_i)$ be the sequence of tree nodes on the path from the root to the word $v_i$ and let $\boldsymbol{\pi}(v_i)$ be the sequence of binary left/right choices for each of those nodes (e.g. $l(v_i)_1$ will always be the root of the tree and $\pi(v_i)_1$ will be 0 if the word leaf node is in its left subtree or 1 otherwise). Let matrix $\mathbf{V}$ now be the matrix containing the logistic regression weights $\mathbf{V}_{l(v_i)_m,:}$ of each tree node $n(v_i)_m$ as its rows and $b_{l(v_i)_m}$ be its bias. The probability $p(v_i = w|\mathbf{v}_{<i})$ is now computed from hidden layer $\mathbf{h}_i(\mathbf{v}_{<i})$ as follows:

$$p(v_i = w|\mathbf{v}_{<i}) = \prod_{m=1}^{|\boldsymbol{\pi}(v_i)|} p(\pi(v_i)_m|\mathbf{v}_{<i}), \quad p(\pi(v_i)_m = 1|\mathbf{v}_{<i}) = \text{sigm}(b_{l(v_i)_m} + \mathbf{V}_{l(v_i)_m,:}\mathbf{h}_i(\mathbf{v}_{<i}))$$
$$(11)$$

The conditionals of Equation 11 let us compute $p(\mathbf{v}) = \prod_i p(v_i = 1|\mathbf{v}_{<i})$ for any document and the parameters $\{\mathbf{b}, \mathbf{c}, \mathbf{W}, \mathbf{V}\}$ can be learned by minimizing the negative data log-likelihood with stochastic gradient descent. Once the model is trained, it can be used to extract a representation from a new document $\mathbf{v}^*$ by computing the value of its hidden layer after observing all of its words, which we note $\mathbf{h}(\mathbf{v}^*) = \text{sigm}(\mathbf{c} + \sum_i W_{:,v_i^*})$.

For a full binary tree of all $V$ words, computing Equation 11 will involve $O(\log(V))$ binary logistic regressions. In our experiments, we used a randomly generated full binary tree with $V$ leaves, each assigned to a unique word of the vocabulary. An even better option would be to derive the tree using Hoffman coding, which would reduce even more the average path lengths.

Since the computation of each logistic regression is in $O(H)$ and there are $D$ words in a document, the complexity of computing all $p(v_i = w|\mathbf{v}_{<i})$ given the hidden layers is in $O(\log(V)DH)$. The total complexity of computing $p(\mathbf{v})$ and updating the parameters under the model is therefore $O(\log(V)DH + DH)$. When compared to the complexity $O(KVH + DH)$ of Replicated Softmax, this is quite competitive[1]. Indeed, Salakhutdinov and Hinton [2] suggest gradually increasing $K$ from 1 to 25, which is larger than $\log(V)$ for a very large vocabulary of one million words. Also, the number of words in a document $D$ will usually be much smaller than the vocabulary size $V$.

The final model, which we refer to as Document NADE (DocNADE), is illustrated in Figure 1. A pseudocode for computing $p(\mathbf{v})$ and the parameter learning gradients for a given document is provided in the supplementary material and our code is available here: `http://www.dmi.usherb.ca/~larocheh/code/DocNADE.zip`.

## 4.1 Training from bags of word counts

So far, we have assumed that the ordering of the words in the document was known. However, document data sets often take the form of set of word counts vectors in which the original word

order, which is required by DocNADE to specify the sequence of conditionals $p(v_i|\mathbf{v} < i)$, has been lost.

One solution is to assume the following generative story: first, a *seed* document $\widetilde{\mathbf{v}}$ is sampled from DocNADE and, finally, a random permutation of its words is taken to produce the observed document $\mathbf{v}$. This translates into the following probability distribution:

$$p(\mathbf{v}) = \sum_{\widetilde{\mathbf{v}} \in \mathcal{V}(\mathbf{v})} p(\mathbf{v}|\widetilde{\mathbf{v}})p(\widetilde{\mathbf{v}}) = \frac{1}{|\mathcal{V}(\mathbf{v})|} \sum_{\widetilde{\mathbf{v}} \in \mathcal{V}(\mathbf{v})} p(\widetilde{\mathbf{v}}) \tag{12}$$

where $p(\widetilde{\mathbf{v}})$ is modeled by DocNADE and $\mathcal{V}(\mathbf{v})$ is the set of all documents $\widetilde{\mathbf{v}}$ with the same word count vector $\mathbf{n}(\mathbf{v}) = \mathbf{n}(\widetilde{\mathbf{v}})$. This distribution is a mixture over all possible permutations that could have generated the original document $\mathbf{v}$. Now, we can use the fact that sampling uniformly from $\mathcal{V}(\mathbf{v})$ can be done solely on the basis of the word counts of $\mathbf{v}$, by randomly sampling words without replacement from those word counts. Therefore, we can train DocNADE on those generated word sequences, as if they were the original documents from which the word counts were extracted. While this is only an approximation of true maximum likelihood learning on the original documents, we've found it to work well in practice.

This approach of training DocNADE can be understood as learning a model that is good at predicting which new words should be inserted in a document at any position, while maintaining its general semantics. The model is therefore learning not to insert "intruder" words, i.e. words that do not belong with the others. After training, a document's learned representation $\mathbf{h}(\mathbf{v})$ should contain valuable information to identify intruder words for this document. It's interesting to note that the detection of such intruder words has been used previously as a task in user studies to evaluate the quality of the topics learned by LDA, though at the level of single topics and not whole documents [8].

## 5   Related Work

We mentioned that the Replicated Softmax models the distribution over words as a product of topic-dependent factors. The Sparse Additive Generative Model (SAGE) [9] is also based on topic-dependent factors, as well as a background factor. The distribution of a word is the renormalized product of its topic factor and the background factor. Unfortunately, much like the Replicated Softmax, training in SAGE scales linearly with the vocabulary size, instead of logarithmically as in DocNADE. Recent work has also been able to improve the complexity of RBM training on word observations. However, for the specific case of the Replicated Softmax, the proposed method does not allow to remove the linear dependence on $V$ of the complexity [10].

There has been fairly little work on using neural networks to learn generative topic models of documents. Glorot et al. [11], Dauphin et al. [12] have trained neural network autoencoders on documents in their binary bag of words representation, but such neural networks are not generative models of documents. One potential advantage of having a proper generative model under which $p(\mathbf{v})$ can be computed exactly is it becomes possible to do Bayesian learning of the parameters, even on a large scale, using recent online Bayesian inference approaches [13, 14].

## 6   Experiments

We present two quantitative comparison of DocNADE with the Replicated Softmax. The first compares the performance of DocNADE as a generative model, while the later evaluates whether DocNADE hidden layer can be used as a meaningful representation for documents. Following Salakhutdinov and Hinton [2], we use a hidden layer size of $H = 50$ in all experiments. A validation set is always set aside to perform model selection of other hyper-parameters, such as the learning rate and the number of learning passes over the training set (based on early stopping). We also tested the use of a hidden layer hyperbolic tangent nonlinearity $\tanh(x) = (\exp(x) - \exp(-x))/(\exp(x) + \exp(-x))$ instead of the sigmoid and always used the best option based on the validation set performance. We end this section with a qualitative inspection of the implicit word representation and topic-features learned by DocNADE.

| Data Set | LDA (50) | LDA (200) | Replicated Softmax (50) | DocNADE (50) | DocNADE St. Dev |
|---|---|---|---|---|---|
| 20 Newsgroups | 1091 | 1058 | 953 | **896** | 6.9 |
| RCV1-v2 | 1437 | 1142 | 988 | **742** | 4.5 |

Table 1: Test perplexity per word for LDA with 50 and 200 latent topics, Replicated Softmax with 50 topics and DocNADE with 50 topics. The results for LDA and Replicated Softmax were taken from Salakhutdinov and Hinton [2].

## 6.1 Generative Model Evaluation

We first evaluated DocNADE's performance as a generative model of documents. We performed our evaluation on the 20 Newsgroups and the Reuters Corpus Volume I (RCV1-v2) data sets and we followed the same evaluation as in Salakhutdinov and Hinton [2]: word counts were replaced by $\log(1 + n_i)$ rounded to the closest integer and a subset of 50 test documents (2193 words for 20 Newsgroups, 4716 words for RCV1-v2) were used to estimate the test perplexity per word $\exp(-\frac{1}{N} \sum_t \frac{1}{|\mathbf{v}^t|} \log p(\mathbf{v}^t))$. The vocabulary size for 20 Newsgroups was 2000 and 10 000 for RCV1-v2.

We used the version of DocNADE that trains from document word counts. To approximate the corresponding distribution $p(\mathbf{v})$ of Equation 12, we sample a single permuted word sequence $\widetilde{\mathbf{v}}$ from the word counts. This might seem like a crude approximation, but, as we'll see, the value of $p(\widetilde{\mathbf{v}})$ tends not to vary a lot across different random permutations of the words.

Instead of minimizing the average document negative log-likelihood $-\frac{1}{N} \sum_t \log p(\mathbf{v}^t)$, we also considered minimizing a version normalized by each document's size $-\frac{1}{N} \sum_t \frac{1}{|\mathbf{v}^t|} \log p(\mathbf{v}^t)$, though the difference in performance between both ended up not being large. For 20 newsgroups, the model with the best perplexity on the validation set used a learning rate of 0.001, sigmoid hidden activation and optimized the average document negative log-likelihood (non-normalized). For RCV1-v2, a learning rate of 0.1, with sigmoid hidden activation and optimization of the objective normalized by each document's size performed best.

The results are reported in Table 1. A comparison is made with LDA using 50 or 200 topics and the Replicated Softmax with 50 topics. The results for LDA and Replicated Softmax were taken from Salakhutdinov and Hinton [2]. We see that DocNADE achieves lower perplexity than both models. On RCV1-v2, DocNADE reaches a perplexity that is almost half that of LDA with 50 topics. We also provide the standard deviation of the perplexity obtained by repeating 100 times the calculation of the perplexity on the test set using different permuted word sequences $\widetilde{\mathbf{v}}$. We see that it is fairly small, which confirms that the value of $p(\widetilde{\mathbf{v}})$ does not vary a lot across different permutations. This is consistent with the observation made by Larochelle and Murray [3] that results are stable with respect to the choice of ordering for the conditionals $p(v_i|\mathbf{v}_{<i})$.

## 6.2 Document Retrieval Evaluation

We also evaluated the quality of the document representation $\mathbf{h}(\mathbf{v})$ learned by DocNADE in an information retrieval task using the 20 Newsgroups data set and its label information. In this context, all test documents were each used as queries and compared to a fraction of the closest documents in the original training set. Similarity between documents is computed using the cosine angle between document representations. We then compute the average number of retrieved training documents sharing the same label as the query (precision), and so for different fractions of retrieved documents.

For learning, we set aside 1000 documents for validation. For model selection, we used the validation set as the query set and used the average precision at 0.02% retrieved documents as the performance measure. We used only the training objective normalized by the document size and set the maximum number of training passes to 973 (approximately 10 million parameter updates). The best learning rate was 0.01, with tanh hidden activation. Notice that the labels are not used during training. Since Salakhutdinov and Hinton [2] showed that it strictly outperforms LDA on this problem, we only compare to the Replicated Softmax. We performed stochastic gradient descent based on the contrastive divergence approximation during 973 training passes, and so for different learning rates. As recommended in Salakhutdinov and Hinton [2], we gradually increased the num-

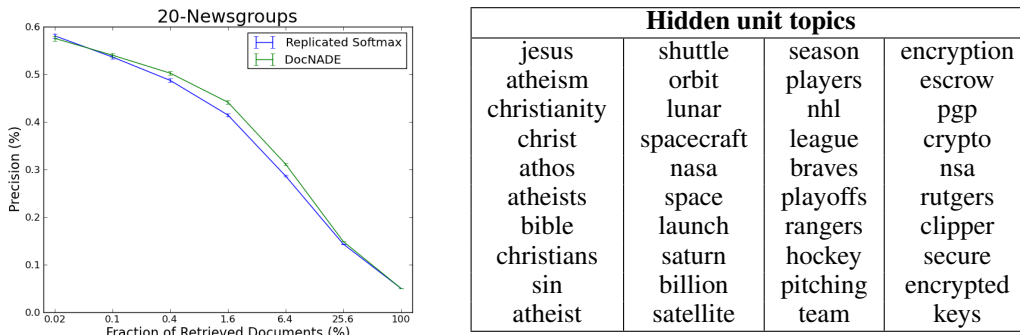

| **Hidden unit topics** | | | |
|---|---|---|---|
| jesus | shuttle | season | encryption |
| atheism | orbit | players | escrow |
| christianity | lunar | nhl | pgp |
| christ | spacecraft | league | crypto |
| athos | nasa | braves | nsa |
| atheists | space | playoffs | rutgers |
| bible | launch | rangers | clipper |
| christians | saturn | hockey | secure |
| sin | billion | pitching | encrypted |
| atheist | satellite | team | keys |

Figure 2: **(Left)** Information retrieval task results, on 20 Newsgroups data set. The error bars correspond to the standard errors. **(Right)** Illustration of some topics learned by DocNADE. A topic $i$ is visualized by picking the 10 words $w$ with strongest connection $W_{iw}$.

Table 2: The five nearest neighbors in the word representation space learned by DocNADE.

| **weapons** | **medical** | **companies** | **define** | **israel** | **book** | **windows** |
|---|---|---|---|---|---|---|
| weapon | treatment | demand | defined | israeli | reading | dos |
| shooting | medecine | commercial | definition | israelis | read | microsoft |
| firearms | patients | agency | refer | arab | books | version |
| assault | process | company | make | palestinian | relevent | ms |
| armed | studies | credit | examples | arabs | collection | pc |

ber of Gibbs sampling steps $K$ from 1 to 25, but also tried increasing it only to 5 or maintaining it to $K = 1$. Optionally, we also used mean-field inference for the first few training passes. The best combination of these choices was selected based on validation performance.

The final results are presented in Figure 2. We see that DocNADE compares favorably with the Replicated Softmax. DocNADE is never outperformed by the Replicated Softmax and outperforms it for the intermediate retrieval fractions.

## 6.3 Qualitative Inspection of Learned Representations

Since topic models are often used for the exploratory analysis of unlabeled text, we looked at whether meaningful semantics were captured by DocNADE. First, to inspect the nature of topics modeled by the hidden units, we looked at the words with strongest positive connections to that hidden unit, i.e. the words $w$ that have the largest values of $W_{i,w}$ for the $i^{\text{th}}$ hidden unit. Figure 2 shows four topics extracted this way and that could be understood as topics about religion, space, sports and security, which are label (sub)categories in 20 Newsgroups. We can also extract word representations, by using the columns $W_{:,w}$ as the vector representation of each word $w$. Table 2 shows the five nearest neighbors of some selected words in this space, confirming that the word representations are meaningful. In the supplementary material, we also provide 2D visualizations of these representations based on t-SNE [15], for 20 Newsgroups and RCV1-v2.

## 7 Conclusion

We have proposed DocNADE, an unsupervised neural network topic model of documents and have shown that it is a competitive model both as a generative model and as a document representation learning algorithm. Its training has the advantageous property of scaling sublinearly with the vocabulary size. Since the early work on topic modeling, research on the subject has progressed by developing Bayesian algorithms for topic modeling, by exploiting labeled data and by incorporating more structure within the latent topic representation. We feel like this is a plausible and most natural course to follow for future research.

## Acknowledgment

We thank Ruslan Salakhutdinov for providing us with the data sets used in the experiments. This work was supported by NSERC and Google.

## Footnotes

[1]In our experiments, a single training pass of DocNADE on the 20 Newgroups and RCV1-v2 data sets (see Section 6.1 for details) took on average 13 seconds and 726 seconds respectively. On the other hand, for $K = 1$ Gibbs sampling steps, our implementation of Replicated Softmax requires 28 seconds and 4945 seconds respectively. For $K = 5$, running time increases even more, to 60 seconds and 11000 seconds.

# References

[1] David M. Blei, Andrew Y. Ng, and Michael I. Jordan. Latent Dirichlet Allocation. *Journal of Machine Learning Research*, 3(4-5):993–1022, 2003.

[2] Ruslan Salakhutdinov and Geoffrey Hinton. Replicated Softmax: an Undirected Topic Model. In *Advances in Neural Information Processing Systems 22 (NIPS 2009)*, pages 1607–1614, 2009.

[3] Hugo Larochelle and Ian Murray. The Neural Autoregressive Distribution Estimator. In *Proceedings of the 14th International Conference on Artificial Intelligence and Statistics (AISTATS 2011)*, volume 15, pages 29–37, Ft. Lauderdale, USA, 2011. JMLR W&CP.

[4] Yoshua Bengio and Samy Bengio. Modeling High-Dimensional Discrete Data with Multi-Layer Neural Networks. In *Advances in Neural Information Processing Systems 12 (NIPS 1999)*, pages 400–406. MIT Press, 2000.

[5] Geoffrey E. Hinton. Training products of experts by minimizing contrastive divergence. *Neural Computation*, 14:1771–1800, 2002.

[6] Frederic Morin and Yoshua Bengio. Hierarchical Probabilistic Neural Network Language Model. In *Proceedings of the 10th International Workshop on Artificial Intelligence and Statistics (AISTATS 2005)*, pages 246–252. Society for Artificial Intelligence and Statistics, 2005.

[7] Andriy Mnih and Geoffrey E Hinton. A Scalable Hierarchical Distributed Language Model. In *Advances in Neural Information Processing Systems 21 (NIPS 2008)*, pages 1081–1088, 2009.

[8] Jonathan Chang, Jordan Boyd-Graber, Sean Gerrish, Chong Wang, and David Blei. Reading Tea Leaves: How Humans Interpret Topic Models. In *Advances in Neural Information Processing Systems 22 (NIPS 2009)*, pages 288–296, 2009.

[9] Jacob Eisenstein, Amr Ahmed, and Eric P. Xing. Sparse Additive Generative Models of Text. In *Proceedings of the 28th International Conference on Machine Learning (ICML 2011)*, pages 1041–1048. Omnipress, 2011.

[10] George E. Dahl, Ryan P. Adams, and Hugo Larochelle. Training Restricted Boltzmann Machines on Word Observations. In *Proceedings of the 29th International Conference on Machine Learning (ICML 2012)*, 2012.

[11] Xavier Glorot, Antoine Bordes, and Yoshua Bengio. Domain Adaptation for Large-Scale Sentiment Classification: A Deep Learning Approach. In *Proceedings of the 28th International Conference on Machine Learning (ICML 2011)*, pages 513–520. Omnipress, 2011.

[12] Yann Dauphin, Xavier Glorot, and Yoshua Bengio. Large-Scale Learning of Embeddings with Reconstruction Sampling. In *Proceedings of the 28th International Conference on Machine Learning (ICML 2011)*, pages 945–952. Omnipress, 2011.

[13] Max Welling and Yee Whye Teh. Bayesian Learning via Stochastic Gradient Langevin Dynamics. In *Proceedings of the 28th International Conference on Machine Learning (ICML 2011)*, pages 681–688. Omnipress, 2011.

[14] Sungjin Ahn, Anoop Korattikara, and Max Welling. Bayesian Posterior Sampling via Stochastic Gradient Fisher Scoring. In *Proceedings of the 29th International Conference on Machine Learning (ICML 2012)*, 2012.

[15] Laurens van der Maaten and Geoffrey E Hinton. Visualizing Data using t-SNE. *Journal of Machine Learning Research*, 9:2579–2605, 2008. URL `http://www.jmlr.org/papers/volume9/vandermaaten08a/vandermaaten08a.pdf`.

